# Design of experiments via information theory [*]

**Liam Paninski**
Center for Neural Science
New York University
New York, NY 10003
*liam@cns.nyu.edu*

## Abstract

We discuss an idea for collecting data in a relatively efficient manner. Our point of view is Bayesian and information-theoretic: on any given trial, we want to adaptively choose the input in such a way that the mutual information between the (unknown) state of the system and the (stochastic) output is maximal, given any prior information (including data collected on any previous trials). We prove a theorem that quantifies the effectiveness of this strategy and give a few illustrative examples comparing the performance of this adaptive technique to that of the more usual nonadaptive experimental design. For example, we are able to explicitly calculate the asymptotic relative efficiency of the "staircase method" widely employed in psychophysics research, and to demonstrate the dependence of this efficiency on the form of the "psychometric function" underlying the output responses.

## 1 Introduction

One simple model of experimental design (we have neurophysiological experiments in mind, but our results are general with respect to the identity of the system under study) is as follows. We have some set $X$ of input stimuli, and some knowledge of how the system should respond to every stimulus, $x$, in $X$. This knowledge is summarized in the form of a prior distribution, $p_0(\theta)$, on some space $\Theta$ of models $\theta$. A model is a set of probabilistic input-output relationships: regular conditional distributions $p(y|x, \theta)$ on $Y$, the set of possible output responses, given each $x$ in $X$. Thus the joint probability of stimulus and response is:

$$p(x, y) = \int p(x, y, \theta) d\theta = \int p_0(\theta) p(x) p(y|\theta, x) d\theta.$$

The "design" of an experiment is given by the choice of input probability $p(x)$. We want to design our experiment — choose $p(x)$ — optimally in some sense. One natural idea would be to choose $p(x)$ in such a way that we learn as much as possible about the underlying model, on average. Information theory thus suggests we choose $p(x)$ to optimize the

---

[*]A longer version of this paper, including proofs, has been submitted and is available at http://www.cns.nyu.edu/~liam.

following objective function:

$$I(\{x, y\}; \theta) = \int_{X \times Y \times \Theta} p(x, y, \theta) \log \frac{p(x, y, \theta)}{p(x, y)p(\theta)} \tag{1}$$

where $I(.;.)$ denotes mutual information. In other words, we want to maximize the information provided about $\theta$ by the pair $\{x, y\}$, given our current knowledge of the model as summarized in the posterior distribution given $N$ samples of data:

$$p_N(\theta) = p(\theta | \{x_i, y_i\}_{1 \le i \le N}).$$

Similar ideas have seen application in a wide and somewhat scattered literature; for a partial bibliography, see the longer draft of this paper at http://www.cns.nyu.edu/~liam. Somewhat surprisingly, we have not seen any applications of the information-theoretic objective function (1) to the design of neurophysiological experiments (although see the abstract by [7], who seem to have independently implemented the same idea in a simulation study).

The primary goal of this paper is to elucidate the asymptotic behavior of the *a posteriori* density $p_N(\theta)$ when we choose $x$ according to the recipe outlined above; in particular, we want to compare the adaptive strategy to the more usual case, in which the stimuli are drawn i.i.d. (non-adaptively) from some fixed distribution $p(x)$. Our main result (section 2) states that, under acceptably weak conditions on the models $p(y|\theta, x)$, the information-maximization strategy leads to consistent and efficient estimates of the true underlying model, in a natural sense. We also give a few simple examples to illustrate the applicability of our results (section 3).

## 2   Main Result

First, we note that the problem as posed in the introduction turns out to be slightly easier than one might have expected, because $I(\{x, y\}; \theta)$ is linear in $p(x)$. This, in turn, implies that $p(x)$ must be degenerate, concentrated on the points $x$ where $I$ is maximal. Thus, instead of finding optimal distributions $p(x)$, we need only find optimal inputs $x$, in the sense of maximizing the conditional information between $\theta$ and $y$, given a single input $x$:

$$I(y; \theta | x) \equiv \int_Y \int_\Theta p_N(\theta) p(y|\theta, x) \log \frac{p(y|x, \theta)}{\int_\Theta p_N(\theta) p(y|x, \theta)}.$$

Our main result is a "Bernstein-von Mises" - type theorem [12]. The classical form of this kind of result says, basically, that if the posterior distributions are consistent (in the sense that $p_N(U) \to 1$ for any neighborhood $U$ of the true parameter $\theta_0$) and the relevant likelihood ratios are sufficiently smooth on average, then the posterior distributions $p_N(\theta)$ are asymptotically normal, with easily calculable asymptotic mean and variance. We adapt this result to the present case, where $x$ is chosen according to the information-maximization recipe. It turns out that the hard part is proving consistency (c.f. section 4); we give the basic consistency lemma (interesting in its own right) first, from which the main theorem follows fairly easily.

**Lemma 1 (Consistency).** *Assume the following conditions:*

1. *The parameter space $\Theta$ is compact.*

2. *The loglikelihood $\log p(y|x, \theta)$ is Lipschitz in $\theta$, uniformly in $x$, with respect to some dominating measure on $Y$.*

3. *The prior measure $p_0$ assigns positive measure to any neighborhood of $\theta_0$.*

4. *The maximal divergence $\sup_x D_{KL}(\theta_0; \theta | x)$ is positive for all $\theta \neq \theta_0$.*

*Then the posteriors are consistent: $p_N(U) \to 1$ in probability for any neighborhood $U$ of $\theta_0$.*

**Theorem 2 (Asymptotic normality).** *Assume the conditions of Lemma 1, stengthened as follows:*

1. *$\Theta$ has a smooth, finite-dimensional manifold structure in a neighborhood of $\theta_0$.*

2. *The loglikelihood $\log p(y|x,\theta)$ is uniformly $C^2$ in $\theta$. In particular, the Fisher information matrices*

$$I_\theta(x) = \int_Y \left( \frac{\dot{p}(y|x,\theta)}{p(y|x,\theta)} \right)^t \left( \frac{\dot{p}(y|x,\theta)}{p(y|x,\theta)} \right) p(y|\theta,x),$$

*where the differential $\dot{p}$ is taken with respect to $\theta$, are well-defined and continuous in $\theta$, uniformly in $(x,\theta)$ in some neighborhood of $\theta_0$.*

3. *The prior measure $p_0$ is absolutely continuous in some neighborhood of $\theta_0$, with a continuous positive density at $\theta_0$.*

4.
$$\max_{C \in \mathrm{co}(I_{\theta_0}(x))} \det(C) > 0,$$

*where $\mathrm{co}(I_{\theta_0}(x))$ denotes the convex closure of the set of Fisher information matrices $I_{\theta_0}(x)$.*

*Then*
$$||p_N - \mathcal{N}(\mu_N, \sigma_N^2)|| \to 0$$

*in probability, where $||.||$ denotes variation distance, $\mathcal{N}(\mu_N, \sigma_N^2)$ denotes the normal density with mean $\mu_N$ and variance $\sigma_N^2$, and $\mu_N$ is asymptotically normal with mean $\theta_0$ and variance $\sigma_N^2$. Here*

$$(N\sigma_N^2)^{-1} \to \mathrm{argmax}_{C \in \mathrm{co}(I_{\theta_0}(x))} \det(C);$$

*the maximum in the above expression is well-defined and unique.*

Thus, under these conditions, the information maximization strategy works, and works better than the i.i.d. $x$ strategy (where the asymptotic variance $\sigma^2$ is inversely related to an average, not a maximum, over $x$, and is therefore generically larger).

A few words about the assumptions are in order. Most should be fairly self-explanatory: the conditions on the priors, as usual, are there to ensure that the prior becomes irrelevant in the face of sufficient posterior evidence; the smoothness assumptions on the likelihood permit the local expansion which is the source of asymptotic normality; and the condition on the maximal divergence function $\sup_x D_{KL}(\theta_0; \theta|x)$ ensures that distinct models $\theta_0$ and $\theta$ are identifiable. Finally, some form of monotonicity or compactness on $\Theta$ is necessary, mostly to bound the maximal divergence function $\sup_x D_{KL}(\theta_0; \theta|x)$ and its inverse away from zero (the lower bound, again, is to ensure identifiability; the necessity of the upper bound, on the other hand, will become clear in section 4); also, compactness is useful (though not necessary) for adapting certain Glivenko-Cantelli bounds [12] for the consistency proof.

It should also be clear that we have not stated the results as generally as possible; we have chosen instead to use assumptions that are simple to understand and verify, and to leave the technical generalizations to the interested reader. Our assumptions should be weak enough for most neurophysiological and psychophysical situations, for example, by assuming that parameters take values in bounded (though possibly large) sets and that tuning curves are not infinitely steep. The proofs of these three results are basically elaborations on Wald's consistency method and Le Cam's approach to the Bernstein-von Mises theorem [12].

# 3 Applications

## 3.1 Psychometric model

As noted in the introduction, psychophysicists have employed versions of the information-maximization procedure for some years [14, 9, 13, 6]. References in [13], for example, go back four decades, and while these earlier investigators usually couched their discussion in terms of variance instead of entropy, the basic idea is the same (note, for example, that minimizing entropy is asymptotically equivalent to minimizing variance, by our main theorem). Our results above allow us to precisely quantify the effectiveness of this stategy.

The standard psychometric model is as follows. The response space $Y$ is binary, corresponding to subjective "yes" or "no" detection responses. Let $f$ be "sigmoidal": a uniformly smooth, monotonically increasing function on the line, such that $f(0) = 1/2$, $\lim_{t \to -\infty} f(t) = 0$ and $\lim_{t \to \infty} f(t) = 1$ (this function represents the detection probability when the subject is presented with a stimulus of strength $t$). Let $f_{a,\theta} = f((t-\theta)/a)$; $\theta$ here serves as a location ("threshold") parameter, while $a$ sets the scale (we assume $a$ is known, for now, although of course this can be relaxed [6]). Finally, let $p(x)$ and $p_0(\theta)$ be some fixed sampling and prior distributions, respectively, both with smooth densities with respect to Lebesgue measure on some interval $\Theta$.

Now, for any fixed scale $a$, we want to compare the performance of the information-maximization strategy to that of the i.i.d. $p(x)$ procedure. We have by theorem 2 that the most efficient estimator of $\theta$ is asymptotically unbiased with asymptotic variance

$$\sigma_{info}^2 \approx (N \sup_x I_{\theta_0}(x))^{-1},$$

while the usual calculations show that the asymptotic variance of any efficient estimator based on i.i.d. samples from $p(x)$ is given by

$$\sigma_{iid}^2 \approx (N \int_X dp(x) I_{\theta_0}(x))^{-1};$$

the key point, again, is that $\sigma_{iid}^{-2}$ is an average, while $\sigma_{info}^{-2}$ is a maximum, and hence $\sigma_{iid} \geq \sigma_{info}$, with equality only in the exceptional case that the Fisher information $I_{\theta_0}(x)$ is constant almost surely in $p(x)$.

The Fisher information here is easily calculated here to be

$$I_\theta = \frac{(\dot{f}_{a,\theta})^2}{f_{a,\theta}(1 - f_{a,\theta})}.$$

We can immediately derive two easy but important conclusions. First, there is just one function $f^*$ for which the i.i.d. sampling strategy is as asymptotically efficient as information-maximization strategy; for all other $f$, information maximization is strictly more efficient. The extremal function $f^*$ is obtained by setting $\sigma_{iid} = \sigma_{info}$, implying that $I_{\theta_0}(x)$ is constant a.e. $[p(x)]$, and so $f^*$ is the unique solution of the differential equation

$$\frac{df^*}{dt} = c \bigg( f^*(t)(1 - f^*(t)) \bigg)^{1/2},$$

where the auxiliary constant $c = \sqrt{I_\theta}$ uniquely fixes the scale $a$. After some calculus, we obtain

$$f^*(t) = \frac{\sin(ct) + 1}{2}$$

on the interval $[-\pi/2c, \pi/2c]$ (and defined uniquely, by monotonicity, as 0 or 1 outside this interval). Since the support of the derivative of this function is compact, this result is quite

dependent of the sampling density $p(x)$; if $p(x)$ places any of its mass outside of the interval $[-\pi/2c, \pi/2c]$, then $\sigma_{iid}^2$ is always strictly greater than $\sigma_{info}^2$. This recapitulates a basic theme from the psychophysical literature comparing adaptive and nonadaptive techniques: when the scale of the nonlinearity $f$ is either unknown or smaller than the scale of the i.i.d. sampling density $p(x)$, adaptive techniques are greatly preferable.

Second, a crude analysis shows that, as the scale $a$ of the nonlinearity shrinks, the ratio $\sigma_{iid}^2/\sigma_{info}^2$ grows approximately as $1/a$; this gives quantitative support to the intuition that the sharper the nonlinearity with respect to the scale of the sampling distribution $p(x)$, the more we can expect the information-maximization strategy to help.

### 3.2   Linear-nonlinear cascade model

We now consider a model that has received increasing attention from the neurophysiology community (see, e.g., [8] for some analysis and relevant references). The model is of cascade form, with a linear stage followed by a nonlinear stage: the input space $X$ is a compact subset of $d$-dimensional Euclidean space (take $X$ to be the unit sphere, for concreteness), and the firing rate of the model cell, given input $\vec{x} \in X$, is given by the simple form

$$E(y|\vec{x}, \theta) = f(<\vec{\theta}, \vec{x}>).$$

Here the linear filter $\vec{\theta}$ is some unit vector in $X'$, the dual space of $X$ (thus, the model space $\Theta$ is isomorphic to $X$), while the nonlinearity $f$ is some nonconstant, nonnegative function on $[-1, 1]$. We assume that $f$ is uniformly smooth, to satisfy the conditions of theorem 2; we also assume $f$ is known, although, again, this can be relaxed. The response space $Y$ — the space of possible spike counts, given the stimulus $\vec{x}$ — can be taken to be the nonnegative integers. For simplicity, let the conditional probabilities $p(y|\vec{x}, \theta)$ be parametrized uniquely by the mean firing rate $f(<\vec{\theta}, \vec{x}>)$; the most convenient model, as usual, is to assume that $p(y|\vec{x}, \theta)$ is Poisson with mean $f(<\vec{\theta}, \vec{x}>)$. Finally, we assume that the sampling density $p(x)$ is uniform on the unit sphere (this choice is natural for several reasons, mainly involving symmetry; see, e.g., [2, 8]), and that the prior $p_0(\theta)$ is positive and continuous (and is therefore bounded away from zero, by the compactness of $\Theta$).

The Fisher information for this model is easily calculated as

$$I_\theta(x) = \frac{(f'(<\vec{\theta}, \vec{x}>))^2}{f(<\vec{\theta}, \vec{x}>)} P_{\vec{x}, \theta},$$

where $f'$ is the usual derivative of the real function $f$ and $P_{\vec{x}, \theta}$ is the projection operator corresponding to $\vec{x}$, restricted to the $(d-1)$-dimensional tangent space to the unit sphere at $\theta$. Theorem 2 now implies that

$$\sigma_{info}^2 \approx \left( N \max_{t \in [-1,1]} \frac{f'(t)^2 g(t)}{f(t)} \right)^{-1},$$

while

$$\sigma_{iid}^2 \approx \left( N \int_{[-1,1]} dp(t) \frac{f'(t)^2 g(t)}{f(t)} \right)^{-1},$$

where $g(t) = 1 - t^2$, $p(t)$ denotes the one-dimensional marginal measure induced on the interval $[-1, 1]$ by the uniform measure $p(x)$ on the unit sphere, and $\sigma^2$ in each of these two expressions multiplies the $(d-1)$-dimensional identity matrix.

Clearly, the arguments of subsection 3.1 apply here as well: the ratio $\sigma_{iid}^2/\sigma_{info}^2$ grows roughly linearly in the inverse of the scale of the nonlinearity. The more interesting asymptotics here, though, are in $d$. This is because the unit sphere has a measure concentration

property [11]: as $d \to \infty$, the measure $p(t)$ becomes exponentially concentrated around 0. In fact, it is easy to show directly that, in this limit, $p(t)$ converges in distribution to the normal measure with mean zero and variance $d^{-2}$. The most surprising implication of this result is seen for nonlinearities $f$ such that $f'(0) = 0$, $f(0) > 0$; we have in mind, for example, symmetric nonlinearities like those often used to model complex cells in visual cortex. For these nonlinearities,

$$\frac{\sigma^2_{info}}{\sigma^2_{iid}} = O(d^{-2}) :$$

that is, the information maximization strategy becomes infinitely more efficient than the usual i.i.d. approach as the dimensionality of the spaces $X$ and $\Theta$ grows.

## 4    A Negative Example

Our next example is more negative and perhaps more surprising: it shows how the information-maximation strategy can fail, in a certain sense, if the conditions of the consistency lemma are not met. Let $\Theta$ be multidimensional, with coordinates which are "independent" in a certain sense, and assume the expected information obtained from one coordinate of the parameter remains bounded strictly away from the expected information obtained from one of the other coordinates. For instance, consider the following model:

$$p(1|x) = \begin{cases} .5 & -1 < x \le \theta_{-1}, \\ f_{-1} & \theta_{-1} < x \le 0, \\ .5 & 0 < x \le \theta_1, \\ f_1 & \theta_1 < x \le 1 \end{cases}$$

where $0 \le f_{-1}, f_1 \le 1$,

$$|f_{-1} - .5| > |f_1 - .5|,$$

are known and $-1 < \theta_{-1} < 0$ and $0 < \theta_1 < 1$ are the parameters we want to learn.

Let the initial prior be absolutely continuous with respect to Lebesgue measure; this implies that all posteriors will have the same property. Then, using the inverse cumulative probability transform and the fact that mutual information is invariant with respect to invertible mappings, it is easy to show that the maximal information we can obtain by sampling from the left is strictly greater than the maximal information obtainable from the right, uniformly in $N$. Thus the information-maximization strategy will sample from $x < 0$ forever, leading to a linear information growth rate (and easily-proven consistency) for the left parameter and non-convergence on the right. Compare the performance of the usual i.i.d. approach for choosing $x$ (using any Lebesgue-dominating measure on the parameter space), which leads to the standard root-$N$ rate for both parameters (i.e., is strongly consistent in posterior probability).

Note that this kind of inconsistency problem does not occur in the case of sufficiently smooth $p(y|x, \theta)$, by our main theorem. Thus one way of avoiding this problem would be to fix a finite sampling scale for each coordinate (i.e., discretizing). Below this scale, no information can be extracted; therefore, when the algorithm hits this "floor" for one coordinate, it will switch to the other. However, it is possible to find other examples which show that the lack of consistency is not necessarily tied to the discontinuous nature of the conditional densities.

## 5    Directions

In this paper, we have presented a rigorous theoretical framework for adaptively designing experiments using an information-theoretic objective function. Most importantly, we

have offered some asymptotic results which clarify the effectiveness of adaptive experiment design using the information-theoretic objective function (1); in addition, we expect that our asymptotic approximations should find applications in approximative computational schemes for optimizing stimulus choice during this type of online experiment. For example, our theorem 2 might suggest the use of a mixture-of-Gaussians representation as an efficient approximation for the posteriors $p_N(\theta)$ [5].

It should be clear that we have left several important questions open. Perhaps the most obvious such question concerns the use of non-information theoretic objective functions. It turns out that many of our results apply with only modest changes if the experiment is instead designed to minimize something like the Bayes mean-square error (perhaps defined only locally if $\Theta$ has a nontrivial manifold structure), for example: in this case, the results in sections 3.1 and 3.2 remain completely unchanged, while the statement of our main theorem requires only slight changes in the asymptotic variance formula (see http://www.cns.nyu.edu/~liam). Thus it seems our results here can add very little to any discussion of what objective function is "best" in general.

We briefly describe a few more open research directions below.

## 5.1 "Batch mode" and stimulus dependencies

Perhaps our strongest assumption here is that the experimenter will be able to freely choose the stimuli on each trial. This might be inaccurate for a number of reasons: for example, computational demands might require that experiments be run in "batch mode," with stimulus optimization taking place not after every trial, but perhaps only after each batch of $k$ stimuli, all chosen according to some fixed distribution $p(x)$. Another common situation involves stimuli which vary temporally, for which the system is commonly modelled as responding not just to a given stimulus $x(t)$, but also to all of its time-translates $x(t - \tau)$. Finally, if there is some cost $C(x_0, x_1)$ associated with changing the state of the observational apparatus from the current state $x_0$ to $x_1$, the experimenter may wish to optimize an objective function which incorporates this cost, for example

$$I(y; \theta | x_1) / C(x_0, x_1).$$

Each of these situations is clearly ripe for further study. Here we restrict ourselves to the first setting, and give a simple conjecture, based on the asymptotic results presented above and inspired by results like those of [1, 4, 10]. First, we state more precisely the optimization problem inherent in designing a "batch" experiment: we wish to choose some sequence, $\{x_i\}_{1 \leq i \leq k}$, to maximize

$$I(\{x_i, y_i\}_{1 \leq i \leq k}; \theta);$$

the main difference here is that $\{x_i\}_{1 \leq i \leq k}$ must be chosen *nonadaptively*, i.e., without sequential knowledge of the responses $\{y_i\}$. Clearly, the order of any sequence of optimal $\{x_i\}_{1 \leq i \leq k}$ is irrelevant to the above objective function; in addition, it should be apparent that if no given piece of data $(x, y)$ is too strong (for example, under Lipschitz conditions like those in lemma 1) that any given elements of such an optimal sequence $\{x_i\}_{1 \leq i \leq k}$ should be asymptotically independent. (Without such a smoothness condition — for example, if some input $x$ could definitively decide between some given $\theta_0$ and $\theta_1$ — then no such asymptotic independence statement can hold, since no more than one sample from such an $x$ would be necessary.) Thus, we can hope that we should be able to asymptotically approximate this optimal experiment by sampling in an i.i.d. manner from some well-chosen $p(x)$. Moreover, we can make a guess as to the identity of this putative $p(x)$:

**Conjecture ("Batch" mode).** *Under suitable conditions, the empirical distribution corre-*

*sponding to any optimal sequence $\{x_i\}_{1 \le i \le k}$,*

$$\hat{p}_k(x) \equiv \frac{1}{k} \sum_{i=1}^{k} \delta(x_i),$$

*converges weakly as $k \to \infty$ to S, the convex set of maximizers in $p(x)$ of*

$$E_\theta \log(\det(E_x I_\theta(x))). \tag{2}$$

Expression (2) above is an average over $p(\theta)$ of terms proportional to the negative entropy of the asymptotic Gaussian posterior distribution corresponding to each $\theta$, and thus should be maximized by any optimal approximant distribution $p(x)$. (Note also that expression (2) is concave in $p(x)$, ensuring the tractability of the above maximization.) In fact, it is not difficult, using the results of Clarke and Barron [3] to prove the above conjecture under the conditions like those of Theorem 2, assuming that $X$ is finite (in which case weak convergence is equivalent to pointwise convergence); we leave generalizations for future work.

## Acknowledgements

We thank R. Sussman, E. Simoncelli, C. Machens, and D. Pelli for helpful conversations. This work was partially supported by a predoctoral fellowship from HHMI.

## References

[1] J. Berger, J. Bernardo, and M. Mendoza. *Bayesian Statistics 4*, chapter On priors that maximize expected information, pages 35–60. Oxford University Press, 1989.

[2] E. Chichilnisky. A simple white noise analysis of neuronal light responses. *Network: Computation in Neural Systems*, 12:199–213, 2001.

[3] B. Clarke and A. Barron. Information-theoretic asymptotics of Bayes methods. *IEEE Transactions on Information Theory*, 36:453 – 471, 1990.

[4] B. Clarke and A. Barron. Jeffreys' prior is asymptotically least favorable under entropy risk. *Journal of Statistical Planning Inference*, 41:37–60, 1994.

[5] P. Deignan, P. Meckl, M. Franchek, J. Abraham, and S. Jaliwala. Using mutual information to pre-process input data for a virtual sensor. In *ACC*, number ASME0043 in American Control Conference, 2000.

[6] L. Kontsevich and C. Tyler. Bayesian adaptive estimation of psychometric slope and threshold. *Vision Research*, 39:2729–2737, 1999.

[7] M. Mascaro and D. Bradley. Optimized neuronal tuning algorithm for multichannel recording. Unpublished abstract at http://www.compscipreprints.com/, 2002.

[8] L. Paninski. Convergence properties of some spike-triggered analysis techniques. *Network: Computation in Neural Systems*, 14:437–464, 2003.

[9] D. Pelli. The ideal psychometric procedure. *Investigative Ophthalmology and Visual Science (Supplement)*, 28:366, 1987.

[10] H. R. Scholl. Shannon optimal priors on iid statistical experiments converge weakly to jeffreys' prior. Available at citeseer.nj.nec.com/104699.html, 1998.

[11] M. Talagrand. Concentration of measure and isoperimetric inequalities in product spaces. *Publ. Math. IHES*, 81:73–205, 1995.

[12] A. van der Vaart. *Asymptotic statistics*. Cambridge University Press, Cambridge, 1998.

[13] A. Watson and A. Fitzhugh. The method of constant stimuli is inefficient. *Perception and Psychophysics*, 47:87–91, 1990.

[14] A. Watson and D. Pelli. QUEST: a Bayesian adaptive psychophysical method. *Perception and Psychophysics*, 33:113–120, 1983.
